# Independent Component Analysis for identification of artifacts in Magnetoencephalographic recordings

Ricardo Vigário[1]*, Veikko Jousmäki[2],
Matti Hämäläinen[2], Riitta Hari[2], and Erkki Oja[1]

[1]Lab. of Computer & Info. Science
Helsinki University of Technology
P.O. Box 2200, FIN-02015 HUT, Finland
{Ricardo.Vigario, Erkki.Oja}@hut.fi

[2]Brain Research Unit, Low Temperature Lab.
Helsinki University of Technology
P.O. Box 2200, FIN-02015 HUT, Finland
{veikko, msh, hari}@neuro.hut.fi

## Abstract

We have studied the application of an independent component analysis (ICA) approach to the identification and possible removal of artifacts from a magnetoencephalographic (MEG) recording. This statistical technique separates components according to the kurtosis of their amplitude distributions over time, thus distinguishing between strictly periodical signals, and regularly and irregularly occurring signals. Many artifacts belong to the last category. In order to assess the effectiveness of the method, controlled artifacts were produced, which included saccadic eye movements and blinks, increased muscular tension due to biting and the presence of a digital watch inside the magnetically shielded room. The results demonstrate the capability of the method to identify and clearly isolate the produced artifacts.

## 1 Introduction

When using a magnetoencephalographic (MEG) record, as a research or clinical tool, the investigator may face a problem of extracting the essential features of the neuromagnetic

---

signals in the presence of artifacts. The amplitude of the disturbance may be higher than that of the brain signals, and the artifacts may resemble pathological signals in shape. For example, the heart's electrical activity, captured by the lowest sensors of a whole-scalp magnetometer array, may resemble epileptic spikes and slow waves (Jousmäki and Hari 1996).

The identification and eventual removal of artifacts is a common problem in electroencephalography (EEG), but has been very infrequently discussed in context to MEG (Hari 1993; Berg and Scherg 1994).

The simplest and eventually most commonly used artifact correction method is *rejection*, based on discarding portions of MEG that coincide with those artifacts. Other methods tend to restrict the subject from producing the artifacts (e.g. by asking the subject to fix the eyes on a target to avoid eye-related artifacts, or to relax to avoid muscular artifacts). The effectiveness of those methods can be questionable in studies of neurological patients, or other non-co-operative subjects. In eye artifact canceling, other methods are available and have recently been reviewed by Vigário (1997b) whose method is close to the one presented here, and in Jung et al. (1998).

This paper introduces a new method to separate brain activity from artifacts, based on the assumption that the brain activity and the artifacts are anatomically and physiologically separate processes, and that their independence is reflected in the statistical relation between the magnetic signals generated by those processes.

The remaining of the paper will include an introduction to the independent component analysis, with a presentation of the algorithm employed and some justification of this approach. Experimental data are used to illustrate the feasibility of the technique, followed by a discussion on the results.

## 2   Independent Component Analysis

Independent component analysis is a useful extension of the principal component analysis (PCA). It has been developed some years ago in context with blind source separation applications (Jutten and Herault 1991; Comon 1994). In PCA, the eigenvectors of the signal covariance matrix $C = E\{xx^T\}$ give the directions of largest variance on the input data $x$. The principal components found by projecting $x$ onto those perpendicular basis vectors are uncorrelated, and their directions orthogonal.

However, standard PCA is not suited for dealing with non-Gaussian data. Several authors, from the signal processing to the artificial neural network communities, have shown that information obtained from a second-order method such as PCA is not enough and higher-order statistics are needed when dealing with the more demanding restriction of independence (Jutten and Herault 1991; Comon 1994). A good tutorial on neural ICA implementations is available by Karhunen et al. (1997). The particular algorithm used in this study was presented and derived by Hyvärinen and Oja (1997a, 1997b).

### 2.1   The model

In blind source separation, the original independent sources are assumed to be unknown, and we only have access to their weighted sum. In this model, the signals recorded in an MEG study are noted as $x_k(i)$ ($i$ ranging from 1 to $L$, the number of sensors used, and $k$ denoting discrete time); see Fig. 1. Each $x_k(i)$ is expressed as the weighted sum of $M$

independent signals $s_k(j)$, following the vector expression:

$$\mathbf{x}_k = \sum_{j=1}^{M} \mathbf{a}(j)s_k(j) = \mathbf{A}\mathbf{s}_k, \tag{1}$$

where $\mathbf{x}_k = [x_k(1), \ldots, x_k(L)]^T$ is an $L$-dimensional data vector, made up of the $L$ mixtures at discrete time $k$. The $s_k(1), \ldots, s_k(M)$ are the $M$ zero mean independent source signals, and $\mathbf{A} = [\mathbf{a}(1), \ldots, \mathbf{a}(M)]$ is a mixing matrix independent of time whose elements $a_{ij}$ are the unknown coefficients of the mixtures. In order to perform ICA, it is necessary to have at least as many mixtures as there are independent sources ($L \geq M$). When this relation is not fully guaranteed, and the dimensionality of the problem is high enough, we should expect the first independent components to present clearly the most strongly independent signals, while the last components still consist of mixtures of the remaining signals. In our study, we did expect that the artifacts, being clearly independent from the brain activity, should come out in the first independent components. The remaining of the brain activity (e.g. $\alpha$ and $\mu$ rhythms) may need some further processing.

The mixing matrix $\mathbf{A}$ is a function of the geometry of the sources and the electrical conductivities of the brain, cerebrospinal fluid, skull and scalp. Although this matrix is unknown, we assume it to be constant, or slowly changing (to preserve some local constancy).

The problem is now to estimate the independent signals $s_k(j)$ from their mixtures, or the equivalent problem of finding the separating matrix $\mathbf{B}$ that satisfies (see Eq. 1)

$$\hat{\mathbf{s}}_k = \mathbf{B}\mathbf{x}_k. \tag{2}$$

In our algorithm, the solution uses the statistical definition of fourth-order cumulant or kurtosis that, for the $i$th source signal, is defined as

$$kurt(s(i)) = E\{s(i)^4\} - 3[E\{s(i)^2\}]^2,$$

where $E(s)$ denotes the mathematical expectation of $s$.

## 2.2   The algorithm

The initial step in source separation, using the method described in this article, is whitening, or sphering. This projection of the data is used to achieve the uncorrelation between the solutions found, which is a prerequisite of statistical independence (Hyvärinen and Oja 1997a). The whitening can as well be seen to ease the separation of the independent signals (Karhunen et al. 1997). It may be accomplished by PCA projection: $\mathbf{v} = \mathbf{V}\mathbf{x}$, with $E\{\mathbf{v}\mathbf{v}^T\} = I$. The whitening matrix $\mathbf{V}$ is given by

$$\mathbf{V} = \Lambda^{-1/2}\Xi^T,$$

where $\Lambda = \mathrm{diag}[\lambda(1), \ldots, \lambda(M)]$ is a diagonal matrix with the eigenvalues of the data covariance matrix $E\{\mathbf{x}\mathbf{x}^T\}$, and $\Xi$ a matrix with the corresponding eigenvectors as its columns.

Consider a linear combination $y = \mathbf{w}^T\mathbf{v}$ of a sphered data vector $\mathbf{v}$, with $\|\mathbf{w}\| = 1$. Then $E\{y^2\} = 1$ and $kurt(y) = E\{y^4\} - 3$, whose gradient with respect to $\mathbf{w}$ is $4E\{\mathbf{v}(\mathbf{w}^T\mathbf{v})^3\}$.

Based on this, Hyvärinen and Oja (1997a) introduced a simple and efficient fixed-point algorithm for computing ICA, calculated over sphered zero-mean vectors $\mathbf{v}$, that is able to find one of the rows of the separating matrix $\mathbf{B}$ (noted $\mathbf{w}$) and so identify one independent source at a time — the corresponding independent source can then be found using Eq. 2. This algorithm, a gradient descent over the kurtosis, is defined for a particular $k$ as

*1. Take a random initial vector $\mathbf{w}_0$ of unit norm. Let $l = 1$.*

2. *Let* $\mathbf{w}_l = E\{\mathbf{v}(\mathbf{w}_{l-1}^T\mathbf{v})^3\} - 3\mathbf{w}_{l-1}$. *The expectation can be estimated using a large sample of* $\mathbf{v}_k$ *vectors (say, 1,000 vectors).*

3. *Divide* $\mathbf{w}_l$ *by its norm (e.g. the Euclidean norm* $\|\mathbf{w}\| = \sqrt{\sum_i \mathbf{w}_i^2}$ *).*

4. *If* $|\mathbf{w}_l^T\mathbf{w}_{l-1}|$ *is not close enough to 1, let* $l = l+1$ *and go back to step 2. Otherwise, output the vector* $\mathbf{w}_l$.

In order to estimate more than one solution, and up to a maximum of $M$, the algorithm may be run as many times as required. It is, nevertheless, necessary to remove the information contained in the solutions already found, to estimate each time a different independent component. This can be achieved, after the fourth step of the algorithm, by simply subtracting the estimated solution $\hat{s} = \mathbf{w}^T\mathbf{v}$ from the unsphered data $\mathbf{x}_k$. As the solution is defined up to a multiplying constant, the subtracted vector must be multiplied by a vector containing the regression coefficients over each vector component of $\mathbf{x}_k$.

## 3 Methods

The MEG signals were recorded in a magnetically shielded room with a 122-channel whole-scalp Neuromag-122 neuromagnetometer. This device collects data at 61 locations over the scalp, using orthogonal double-loop pick-up coils that couple strongly to a local source just underneath, thus making the measurement "near-sighted" (Hämäläinen et al. 1993).

One of the authors served as the subject and was seated under the magnetometer. He kept his head immobile during the measurement. He was asked to blink and make horizontal saccades, in order to produce typical ocular artifacts. Moreover, to produce myographic artifacts, the subject was asked to bite his teeth for as long as 20 seconds. Yet another artifact was created by placing a digital watch one meter away from the helmet into the shieded room. Finally, to produce breathing artifacts, a piece of metal was placed next to the navel. Vertical and horizontal electro-oculograms (VEOG and HEOG) and electrocardiogram (ECG) between both wrists were recorded simultaneously with the MEG, in order to guide and ease the identification of the independent components. The bandpass-filtered MEG (0.03–90 Hz), VEOG, HEOG, and ECG (0.1–100 Hz) signals were digitized at 297 Hz, and further digitally low-pass filtered, with a cutoff frequency of 45 Hz and downsampled by a factor of 2. The total length of the recording was 2 minutes. A second set of recordings was performed, to assess the reproducibility of the results.

Figure 1 presents a subset of 12 spontaneous MEG signals from the frontal, temporal and occipital areas. Due to the dimension of the data (122 magnetic signals were recorded), it is impractical to plot all MEG signals (the complete set is available on the internet — see reference list for the adress (Vigário 1997a)). Also both EOG channels and the electrocardiogram are presented.

## 4 Results

Figure 2 shows sections of 9 independent components (IC's) found from the recorded data, corresponding to a 1 min period, starting 1 min after the beginning of the measurements. The first two IC's, with a broad band spectrum, are clearly due to the musclular activity originated from the biting. Their separation into two components seems to correspond, on the basis of the field patterns, to two different sets of muscles that were activated during the process. IC3 and IC5 are, respectively showing the horizontal eye movements and the eye blinks, respectively. IC4 represents cardiac artifact that is very clearly extracted. In agreement with Jousmäki and Hari (1996), the magnetic field pattern of IC4 shows some predominance on the left.

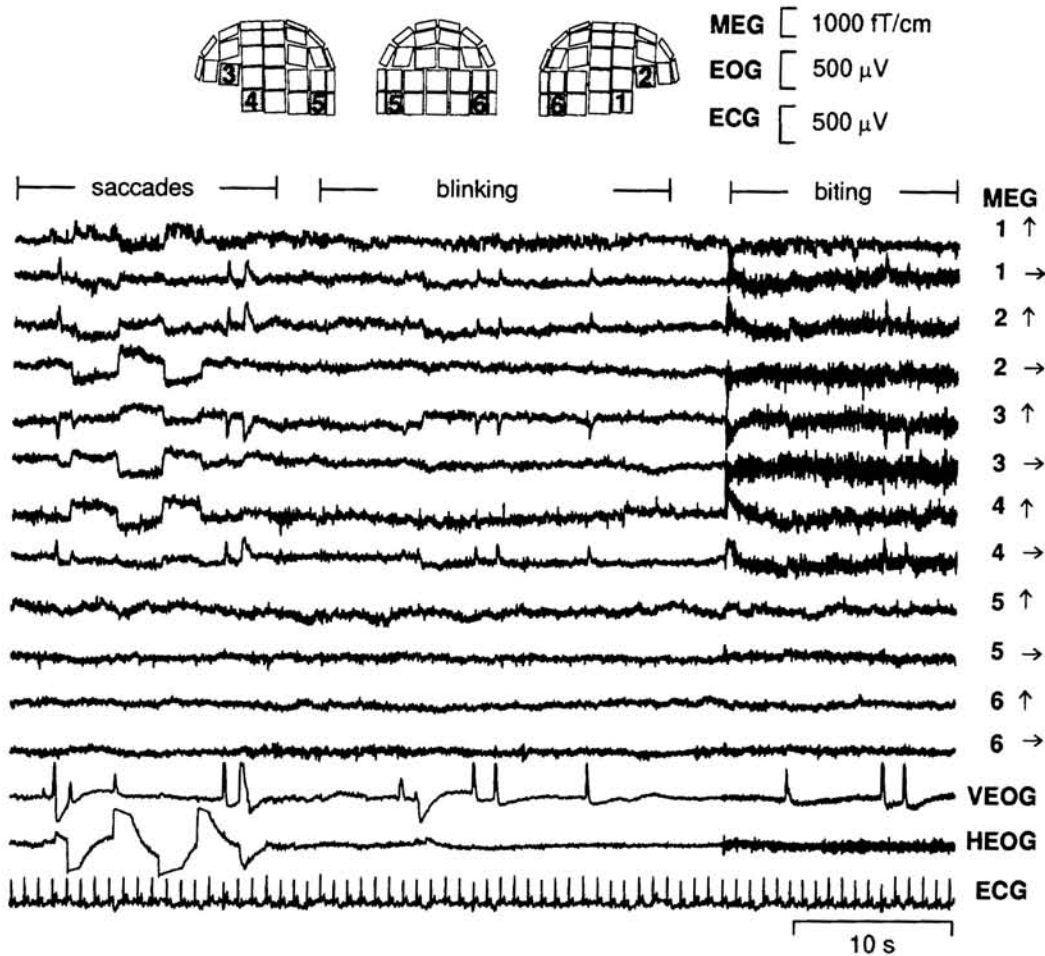

Figure 1: *Samples of MEG signals, showing artifacts produced by blinking, saccades, biting and cardiac cycle. For each of the 6 positions shown, the two orthogonal directions of the sensors are plotted.*

The breathing artifact was visible in several independent components, e.g. IC6 and IC7. It is possible that, in each breathing the relative position and orientation of the metallic piece with respect to the magnetometer has changed. Therefore, the breathing artifact would be associated with more than one column of the mixing matrix **A**, or to a time varying mixing vector.

To make the analysis less sensible to the breathing artifact, and to find the remaining artifacts, the data were high-pass filtered, with cutoff frequency at 1 Hz. Next, the independent component IC8 was found. It shows clearly the artifact originated at the digital watch, located to the right side of the magnetometer.

The last independent component shown, relating to the first minute of the measurement, shows an independent component that is related to a sensor presenting higher RMS (root mean squared) noise than the others.

## 5  Discussion

The present paper introduces a new approach to artifact identification from MEG recordings, based on the statistical technique of Independent Component Analysis. Using this method, we were able to isolate both eye movement and eye blinking artifacts, as well as

cardiac, myographic, and respiratory artifacts.

The basic assumption made upon the data used in the study is that of independence between brain and artifact waveforms. In most cases this independence can be verified by the known differences in physiological origins of those signals. Nevertheless, in some event-related potential (ERP) studies (e.g. when using infrequent or painful stimuli), both the cerebral and ocular signals can be similarly time-locked to the stimulus. This local time dependence could in principle affect these particular ICA studies. However, as the independence between two signals is a measure of the similarity between their joint amplitude distribution and the product of each signal's distribution (calculated throughout the entire signal, and not only close to the stimulus applied), it can be expected that the very local relation between those two signals, during stimulation, will not affect their global statistical relation.

# 6 Acknowledgment

Supported by a grant from Junta Nacional de Investigação Científica e Tecnológica, under its 'Programa PRAXIS XXI' (R.V.) and the Academy of Finland (R.H.).

# References

Berg, P. and M. Scherg (1994). A multiple source approach to the correction of eye artifacts. *Electroenceph. clin. Neurophysiol. 90*, 229–241.

Comon, P. (1994). Independent component analysis - a new concept? *Signal Processing 36*, 287–314.

Hämäläinen, M., R. Hari, R. Ilmoniemi, J. Knuutila, and O. V. Lounasmaa (1993, April). Magnetoencephalography—theory, instrumentation, and applications to noninvasive studies of the working human brain. *Reviews of Modern Physics 65*(2), 413–497.

Hari, R. (1993). Magnetoencephalography as a tool of clinical neurophysiology. In E. Niedermeyer and F. L. da Silva (Eds.), *Electroencephalography. Basic principles, clinical applications, and related fields*, pp. 1035–1061. Baltimore: Williams & Wilkins.

Hyvärinen, A. and E. Oja (1997a). A fast fixed-point algorithm for independent component analysis. *Neural Computation* (9), 1483–1492.

Hyvärinen, A. and E. Oja (1997b). One-unit learning rules for independent component analysis. In *Neural Information Processing Systems 9 (Proc. NIPS'96)*. MIT Press.

Jousmäki, V. and R. Hari (1996). Cardiac artifacts in magnetoencephalogram. *Journal of Clinical Neurophysiology 13*(2), 172–176.

Jung, T.-P., C. Humphries, T.-W. Lee, S. Makeig, M. J. McKeown, V. Iragui, and T. Sejnowski (1998). Extended ica removes artifacts from electroencephalographic recordings. In *Neural Information Processing Systems 10 (Proc. NIPS'97)*. MIT Press.

Jutten, C. and J. Herault (1991). Blind separation of sources, part i: an adaptive algorithm based on neuromimetic architecture. *Signal Processing 24*, 1–10.

Karhunen, J., E. Oja, L. Wang, R. Vigário, and J. Joutsensalo (1997). A class of neural networks for independent component analysis. *IEEE Trans. Neural Networks 8*(3), 1–19.

Vigário, R. (1997a). WWW adress for the MEG data: http://nucleus.hut.fi/~rvigario/NIPS97_data.html.

Vigário, R. (1997b). Extraction of ocular artifacts from eeg using independent component analysis. To appear in *Electroenceph. clin. Neurophysiol.*

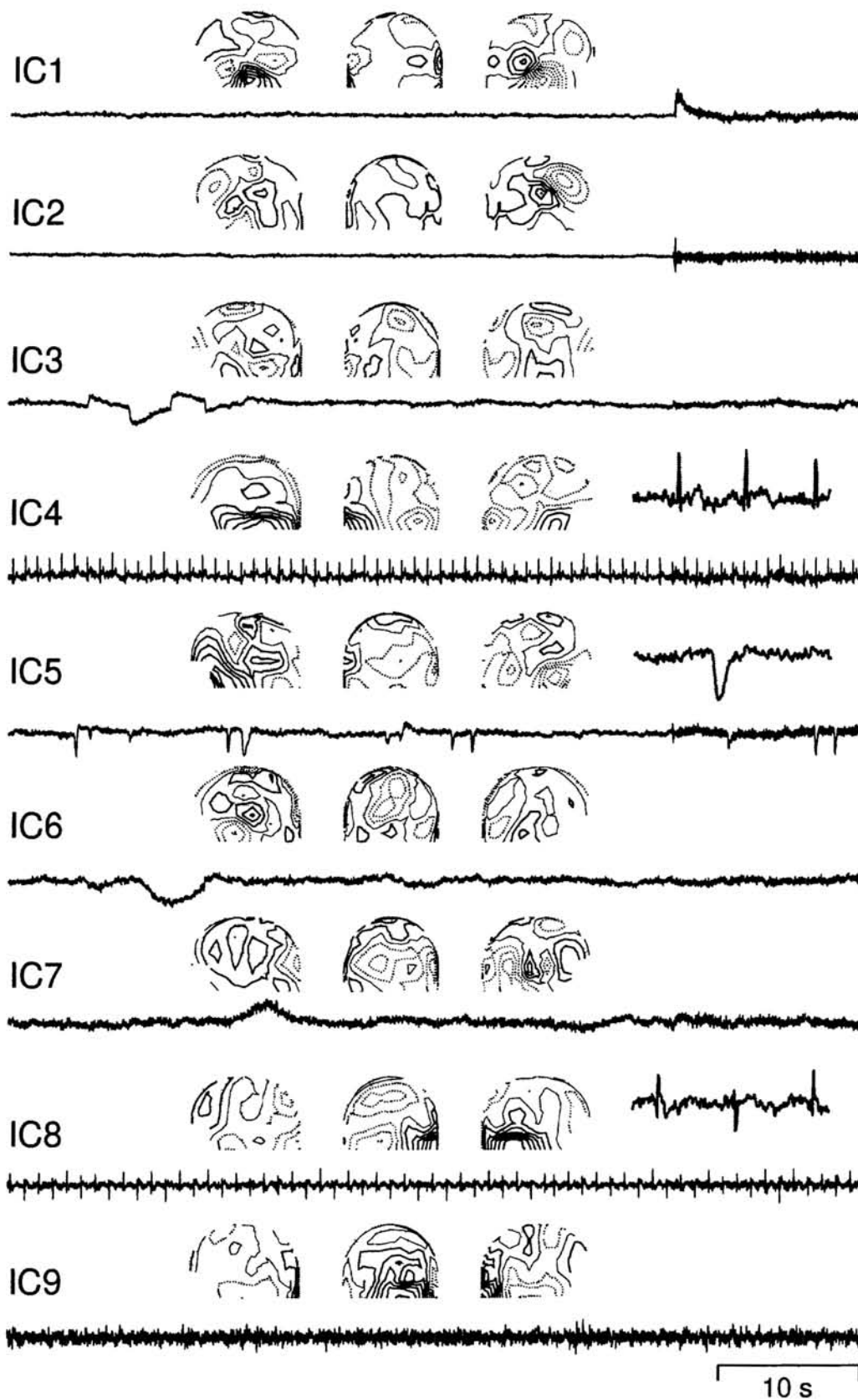

Figure 2: *Nine independent components found from the MEG data. For each component the left, back and right views of the field patterns generated by these components are shown — full line stands for magnetic flux coming out from the head, and dotted line the flux inwards.*